# Silicon Auditory Processors

## as

## Computer Peripherals

John Lazzaro. John Wawrzynek
CS Division
UC Berkeley
Evans Hall
Berkeley, CA 94720
lazzaro@cs.berkeley.edu, johnw@cs.berkeley.edu

M. Mahowald*, Massimo Sivilotti[†], Dave Gillespie[‡]
California Institute of Technology
Pasadena, CA 91125

## Abstract

Several research groups are implementing analog integrated circuit models of biological auditory processing. The outputs of these circuit models have taken several forms, including video format for monitor display, simple scanned output for oscilloscope display and parallel analog outputs suitable for data-acquisition systems. In this paper, we describe an alternative output method for silicon auditory models, suitable for direct interface to digital computers.

# 1. INTRODUCTION

Several researchers have implemented computational models of biological auditory processing, with the goal of incorporating these models into a speech recognition system (for a recent review, see (Jankowski, 1992)). These projects have shown the promise of the biological approach, sometimes showing clear performance advantages over traditional methods.

The application of these computational models is limited by their large computation and communication requirements. Analog VLSI implementations of these neural models may relieve this computational burden; several VLSI research groups have efforts in this area, and working integrated circuit models of many popular representations presently exist. A review of these models is presented in (Lazzaro, 1991). In this paper, we present an interface method (Mahowald, 1992; Sivilotti, 1991) that addresses the communications issues between analog VLSI auditory implementations and digital processors.

# 2. COMMUNICATIONS IN NEURAL SYSTEMS

Biological neurons communicate long distances using a pulse representation. Communications engineers have developed several schemes for communicating on a wire using pulses as atomic units. In these schemes, maximally using the communications bandwidth of a wire implies the mean rate of pulses on the wire is a significant fraction of the maximum pulse rate allowed on the wire.

Using this criterion, neural systems use wires very inefficiently. In most parts of the brain, most of the wires are essentially inactive most of the time. If neural systems are not organized to fully utilize the available bandwidth of each wire, what does neural communication optimize? Evidence suggests that energy conservation is an important issue for neural systems. A simple strategy for energy conservation is the reduction of the total number of pulses in the representation. Many possible coding strategies satisfy this energy requirement.

The strategies observed in neural systems share another common property. Neural systems often implement a class of computations in a manner that produces an energy-efficient output encoding as an additional byproduct. The energy-efficient coding is not performed simply for communication and immediately reversed upon receipt, but is an integral part of the new representation. In this way, energy-efficient neural coding is intrinsically different from engineering data compression techniques.

Temporal adaptation, lateral inhibition, and spike correlations are examples of neural processing methods that perform interesting computations while producing an energy-efficient output code. These representational principles are the foundation of the neural computation and communication method we advocate in this paper. In this method, the output units of a chip are spiking neuron circuits that use energy-efficient coding methods. To communicate this code off a chip, we use a distinctly non-biological approach.

## 3. THE EVENT-ADDRESS PROTOCOL

The unique characteristics of energy-efficient codes define the remaining off-chip communications problem. In the spiking neuron protocol, the height and width of the spike carries no information; the neuron imparts new information only at the moment a spike begins. This moment occurs asynchronously; there is no global clock synchronizing the output units. One way of completely specifying the information in the output units is an event list, a tabulation of the precise time each output unit begins a new spike. We can use this specification as a basis for an off-chip communications system, that sends an event-list message off-chip at the moment an output neuron begins a new spike. An event-list message includes the identification of the output unit, and the time of firing. A performance analysis of this protocol can be found in (Lazzaro *et al.*, 1992).

Note that an explicit timestamp for each entry in the event list is not necessary, if communication latency between the sending chip and the receiver is a constant. In this case, the sender simply communicates, upon onset of a spike from an output, the identity of the output unit; the receiver can append a locally generated timestamp to complete the event. If simplified in this manner, we refer to the event-list protocol as the event-address protocol.

We have designed a working system that computes a model of auditory nerve response, in real time, using analog VLSI processing. This system takes as input an analog sound source, and uses the event-list representation to communicate the model output to the host computer.

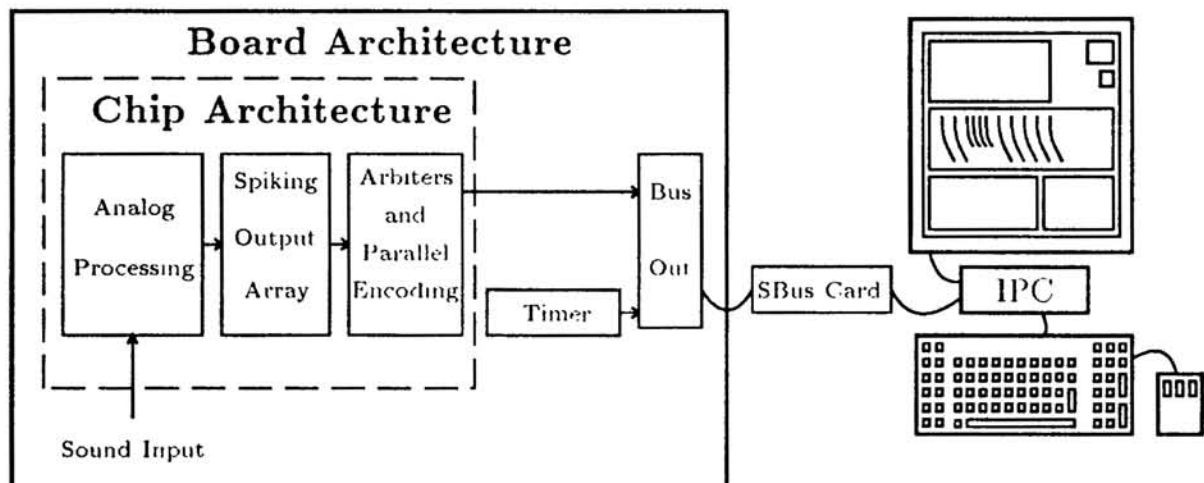

**Figure 1.** System block diagram, showing chip architecture, board architecture, and the host computer (Sun IPC).

## 4. SYSTEMS IMPLEMENTATION

Figure 1 is a block diagram of this system. A single VLSI chip computes the auditory model response; an array of spiking neuron circuits is the final representation of the model. This chip also implements the event-address protocol, using asynchronous arbitration circuits. The chip produces a parallel binary encoding of the model output, as an asynchronous stream of event addresses. These on-chip operations are shown inside the dashed rectangle in Figure 1, labelled **Chip Architecture**.

Additional digital processing completes the custom hardware in the system. This hardware transforms the event-address protocol into an event-list protocol, by adding a time marker for each event (16 bit time markers with $20\mu$s resolution). In addition, the hardware implements the bus interface to the host computer, in conjunction with a commercial interface board. The commercial interface board supports 10 MBytes/second asynchronous data transfers between our custom hardware and the host computer, and includes 8 KBytes of data buffers. Our display software produces a real-time graphical display of the auditory model response, using the X window system.

## 5. VLSI CIRCUIT DETAILS

Figure 2 shows a block diagram of the chip. The analog input signal connects to circuits that perform analog processing, that are fully described and referenced in (Lazzaro *et al.*, 1993). The output of this analog processing is represented by 150 spiking neurons, arranged in a 30 by 5 array. These are the output units of the chip; the event-address protocol communicates the activity of these units off chip. At the onset of a spike from an output unit, the array position of the spiking unit, encoded as a binary number, appears on the output bus. The asynchronous output bus is shown in Figure 2 as the data signals marked **Encoded X Output** (column position) and **Encoded Y Output** (row position), and the acknowledge and request control signals $A_c$ and $R_c$.

We implemented the event-address protocol as an asynchronous arbitration protocol in two dimensions. In this scheme, an output unit can access two request lines, one associated with its row and one associated with its column. Using a wire-OR signalling protocol, any output unit on a particular row or column may assert the request line. Each request line is paired with an acknowledge line, driven by the arbitration circuitry outside the array. Row and column wires for acknowledge and request are explicitly shown in Figure 2, as the lines that form a grid inside the output unit array.

At the onset of a spike, an output unit asserts its row request line, and waits for a reply on its row acknowledge line. An asynchronous arbitration system, marked in Figure 2 as **Y Arbitration Tree**, assures only one output row is acknowledged. After row acknowledgement, the output unit asserts its column request line, and waits for a reply on its column acknowledge line. The arbitration system is shown in detail in Figure 2: four two-input arbiter circuits, shown as rectangles marked with the letter A, are connected as a tree to arbitrate among the 5 column inputs.

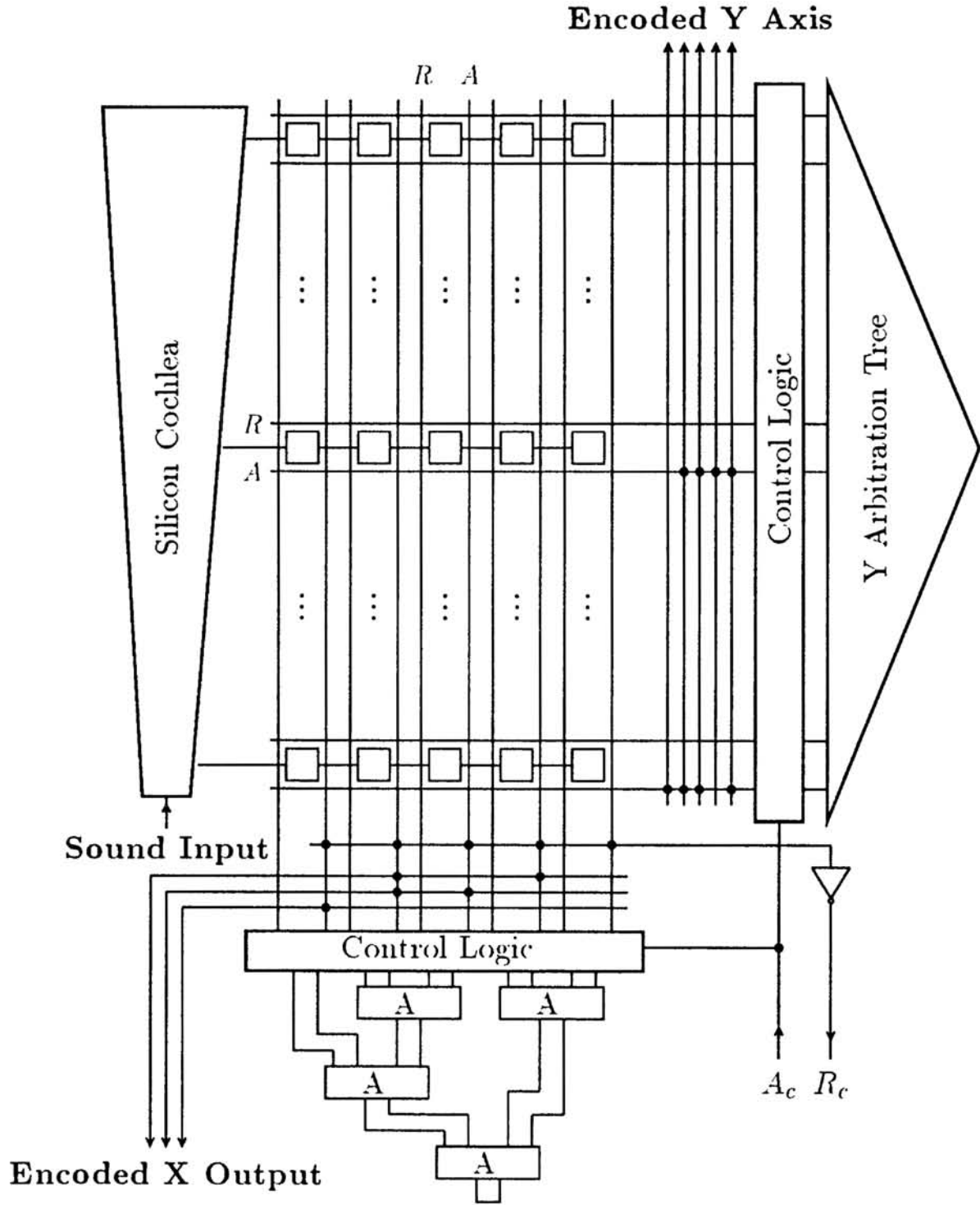

**Figure 2.** Block diagram of the chip. See text for details.

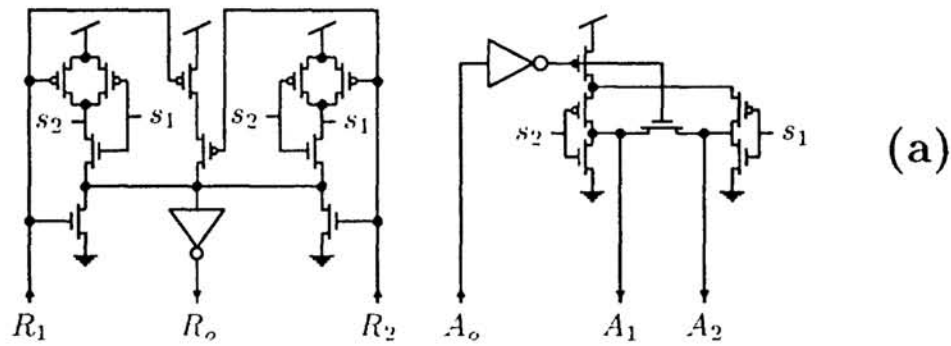

(a)

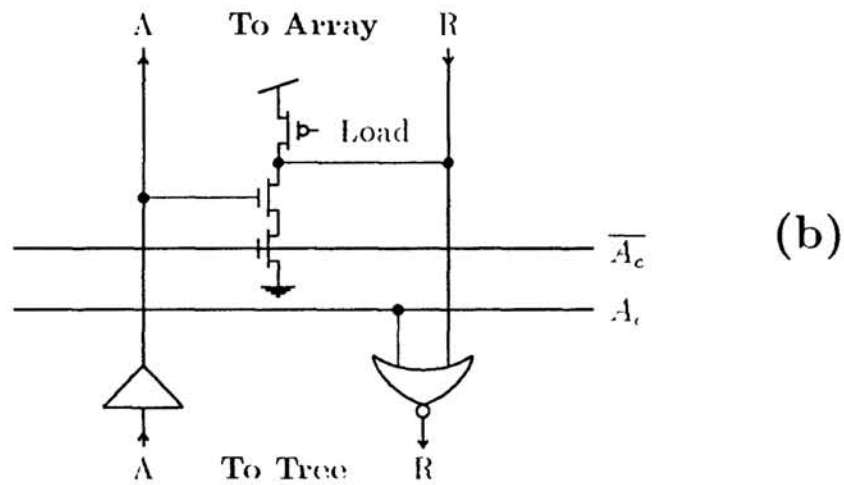

(b)

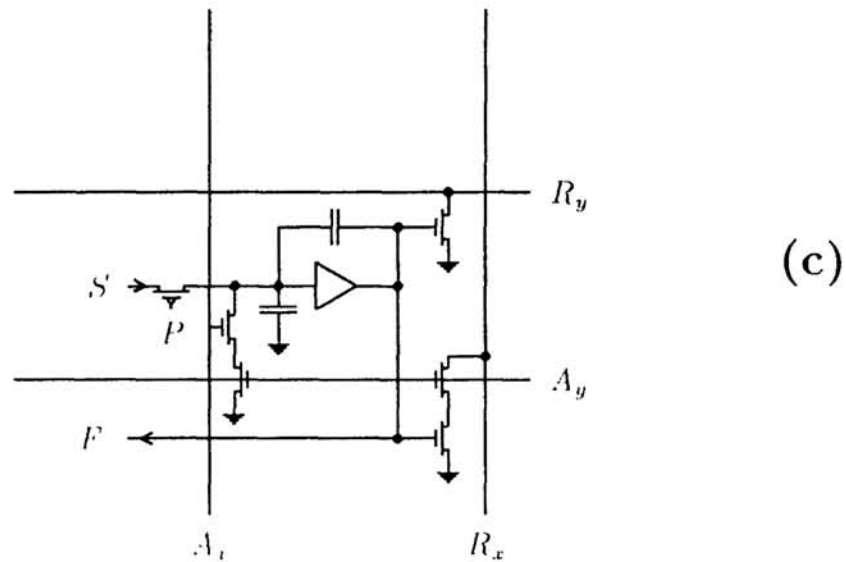

(c)

**Figure 3.** Diagrams of communication circuits in the chip. (a) Two-input arbiter circuit. (b) Control logic to interface arbitration logic and output unit array. (c) Output unit circuit.

Upon the arrival of both row and column acknowledgements, the output unit releases both row and column request lines. Static latches, shown in Figure 2 as the rectangles marked **Control Logic**, retain the state of the row and column request lines.

Binary encoders transform the row and column acknowledge lines into the output data bus. Another column encoder senses the acknowledgement of any column, and asserts the bus control output $R_c$. When the external device has secured the data, it responds by asserting the $A_c$ signal. The $A_c$ signal clears the static latches in the **Control Logic** blocks and resets $R_c$. When $A_c$ is reset, the data transfer is complete, and the chip is ready for the next communication event.

Figure 3 shows the details of the communications circuits of Figure 2. Figure 3(a) shows the two-input arbiter circuit used to create the binary arbitration trees in Figure 3. This digital circuit takes as input two request signals, $R_1$ and $R_2$, and produces the associated acknowledge signals $A_1$ and $A_2$. The acknowledgement of a request precludes the acknowledgement of a second request. The circuit asserts an acknowledge signal until its associated request is released.

$R_o$ is an auxiliary output signal indicating either $R_1$ or $R_2$ has been asserted; $A_o$ is an auxiliary input signal that enables the $A_1$ and $A_2$ outputs. The auxiliary signals allow the two-input arbiter to function as an element in arbitration trees, as shown in Figure 2; the $R_o$ and $A_o$ signals of one level of arbitration connect to the $R_k$ and $A_k$ signals at the next level of arbitration. In two-input operation, the $R_o$ and $A_o$ signals are connected together, as shown in the root arbiter in Figure 2.

Figure 3(b) shows the circuit implementation of the **Control Logic** blocks in Figure 2; this circuit is repeated for each row and column connection. This circuit interfaces the output bus control input $A_c$ with the arbitration circuitry. If output communication is not in progress, $A_c$ is at ground, and $\overline{A_c}$ is at $V_{dd}$.

The PFET transistor marked as **Load** acts as a static pullup to the array request line (R); output units pull this line low to assert a request. The NOR gate inverts the array request line, and routes it to the arbitration tree. When a pending request is acknowledged by the tree acknowledge line, the two NFET transistors act to latch the array request line. The assertion of $A_c$ releases the array request line and disables the arbitration tree request input; these actions reset all state in the communications system. When $A_c$ is released, the system is ready to communicate a new event.

Figure 3(c) shows the circuit implementation of a unit in the output array. In this implementation, each output unit is a two-stage low-power axon circuit (Lazzaro, 1992). The first axonal stage receives the cochlear input; this axon stage is not shown in Figure 3(c). The first stage couples into the second stage, shown in Figure 3(c), via the $S$ and $F$ wires.

To understand the operation of this circuit, we consider the transmission of a single spike. Initially, we assume the request lines $R_x$ and $R_y$ are held high by the static pullup PFET transistors shown in Figure 3(b); in addition, we assume the acknowledge lines $A_x$ and $A_y$ are at ground, and the noninverting buffer input voltage is at

ground.

When the first axonal stage fires, the $S$ signal changes from ground potential to $V_{dd}$. At this point the buffer input voltage begins to increase, at a rate determined by the analog control voltage $P$. When the switching threshold of the buffer is reached, the buffer output voltage $F$ swings to $V_{dd}$; capacitive feedback ensures a reliable switching transition. At this point, the output unit pulls the request line $R_y$ low, and the communications sequence begins.

The Y arbitration logic replies to the $R_y$ request by asserting the $A_y$ line. When both $F$ and $A_y$ are asserted, the output unit pulls the request line $R_x$ low. The X arbitration logic replies to the $R_x$ request by asserting the $A_x$ line. The assertion of both $A_x$ and $A_y$ resets the buffer input voltage to ground. As a result, the $F$ line swings to ground potential, the output unit releases the $R_x$ and $R_y$ lines, and the first axon stage is enabled. At this point, the latch circuit of Figure 3(b) maintains the state of the $R_x$ and $R_y$ lines, until it is cleared by the off-chip acknowledge signal.

## Acknowledgements

Research and prototyping of the event-address interface took place in Carver Mead's laboratory at Caltech; we are grateful for his insights, encouragement, and support. The Caltech-based research was funded by the ONR, HP, and the Systems Development Foundation. Research and prototyping of the auditory-nerve demonstration chip and system took place at UC Berkeley, and was funded by the NSF (PYI award MIPS-895-8568), AT&T, and the ONR (URI-N00014-92-J-1672).

## Footnotes

* Present address: M. Mahowald, MRC Anatomical Neurophamacology Unit, Mansfield Rd, Oxford OX1 3TH England. mam@vax.oxford.ac.uk

† Present address: Mass Sivilotti, Tanner Research, 180 North Vinedo Avenue, Pasadena, CA 91107. mass@tanner.com

‡ Present address: Dave Gillespie, Synaptics, 2698 Orchard Parkway, San Jose CA, 95134. daveg@synaptics.com

## References

Jankowski, C. R. (1992). "A Comparison of Auditory Models for Automatic Speech Recognition," S.B. Thesis, MIT Dept of Electrical Engineering and Computer Science.

Lazzaro, J. P. (1991). "Biologically-based auditory signal processing in analog VLSI," *IEEE Asilomar Conference on Signals, Systems, and Computers.*

Lazzaro, J. P. (1992). "Low-power silicon spiking neurons and axons," *IEEE International Symposium on Circuits and Systems,* San Diego, CA, p. 2220-2224.

Lazzaro, J., Wawrzynek, J., Mahowald, M., Sivilotti, M., and Gillespie, D. (1993). "Silicon auditory processors as computer peripherals," *IEEE Transactions of Neural Networks,* May (in press).

Mahowald, M. (1992). Ph.D. Thesis, Computation and Neural Systems, California Institute of Technology.

Sivilotti, M. (1991). "Wiring considerations in analog VLSI systems, with applications to field-programmable networks," Computer Science Technical Report (Ph. D. Thesis), California Institute of Technology.